# LEARNING THE SOLUTION TO THE APERTURE PROBLEM FOR PATTERN MOTION WITH A HEBB RULE

Martin I. Sereno
Cognitive Science C-015
University of California, San Diego
La Jolla, CA 92093-0115

## ABSTRACT

The primate visual system learns to recognize the true direction of pattern motion using local detectors only capable of detecting the component of motion perpendicular to the orientation of the moving edge. A multilayer feedforward network model similar to Linsker's model was presented with input patterns each consisting of randomly oriented contours moving in a particular direction. Input layer units are granted component direction and speed tuning curves similar to those recorded from neurons in primate visual area V1 that project to area MT. The network is trained on many such patterns until most weights saturate. A proportion of the units in the second layer solve the aperture problem (e.g., show the same direction-tuning curve peak to plaids as to gratings), resembling pattern-direction selective neurons, which first appear in area MT.

## INTRODUCTION

Supervised learning schemes have been successfully used to learn a variety of input-output mappings. Explicit neuron-by-neuron error signals and the apparatus for propagating them across layers, however, are not realistic in a neurobiological context. On the other hand, there is ample evidence in real neural networks for conductances sensitive to correlation of pre- and post-synaptic activity, as well as multiple areas connected by topographic, somewhat divergent feedforward projections. The present project was to try to learn the solution to the aperture problem for pattern motion using a simple hebb rule and a layered feedforward network.

Some of the connections responsible for the selectivity of cortical neurons to local stimulus features develop in the absence of pattered visual experience. For example, newborn cats and primates already have orientation-selective neurons in primary visual cortex (area 17 or V1), before they open their eyes. The prenatally generated orientation selectivity is sharpened by subsequent visual experience. Linsker (1986)

has shown that feedforward networks with somewhat divergent, topographic interlayer connections, linear summation, and simple hebb rules develop units in tertiary and higher layers that have parallel, elongated excitatory and inhibitory subfields when trained solely on random inputs to the first layer.

By contrast, the development of the circuitry in secondary and tertirary visual cortical areas necessary for processing more complex, non-local features of visual arrays--e.g., orientation gradients, shape from shading, pattern translation, dilation, rotation--is probably much more dependent on patterned visual experience. Parietal visual cortical areas, for example, are almost totally unresponsive in dark-reared monkeys, despite the fact that these monkeys have a normal-appearing V1 (Hyvarinen, 1984). Behavioral indices suggest that development of some perceptual abilities may require months of experience. Human babies, for example, only evidence seeing the transition between randomly moving dots and circular 2-D motion at 6 months, while the transition from horizontally moving dots with random x-axis velocities to dots with sinusoidally varying x-axis velocities (the latter gives the percept of a rotating 3-D cylinder) is only detected after 7 months (Spitz, Stiles-Davis, & Siegel, 1988) (see Fig. 1).

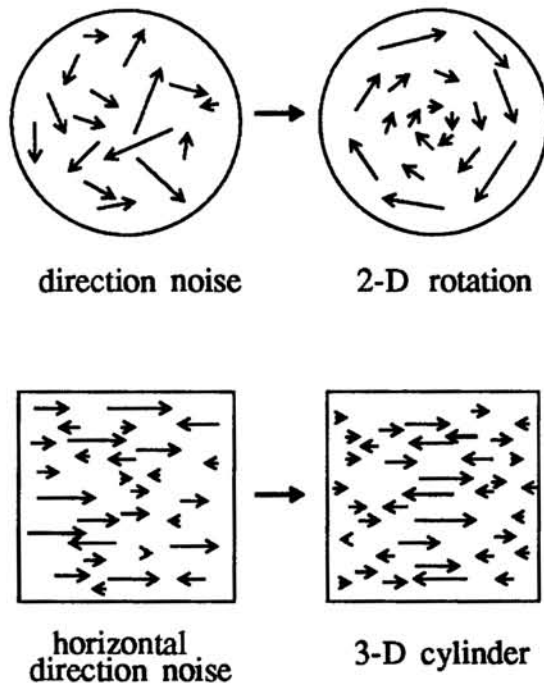

direction noise     2-D rotation

horizontal direction noise     3-D cylinder

**Figure 1.** Motion field transitions

During the first 6 months of its life, a human baby typically makes approximately 30 million saccades, experiencing in the process many views which contain large moving fields and smaller moving objects. The importance of these millions of glances for the development of the ability to recognize complex visual objects has often been acknowledged. Brute visual experience may, however, be just as important in developing a solution to the simpler problem of detecting pattern motion using local cues.

## NETWORK ARCHITECTURE

Moving visual stimuli are processed in several stages in the primate visual system. The first cortical stage is layer 4C-alpha of area V1, which receives its main ascending input from the magnocellular layers of the lateral geniculate nucleus. Layer 4C-alpha projects to layer 4B, which contains many tightly-tuned direction-selective neurons (Movshon et al., 1985). These neurons, however, respond to

moving contours as if these contours were moving perpendicular their local orientation--i.e., they fire in proportion to the difference between the *orthogonal component* of motion and their best direction (for a bar). An orientation series run for a layer 4B neuron using a plaid (2 orthogonal moving gratings) thus results in two peaks in the direction tuning curve, displaced 45 degrees to either side of the peak for a single grating (Movshon et al., 1985). The aperture problem for pattern motion (see e.g., Horn & Schunck, 1981) thus exists for cells in area V1 of the adult (and presumably infant) primate.

Layer 4B neurons project topographically via direct and indirect pathways to area MT, a small extrastriate area specialized for processing moving stimuli. A subset

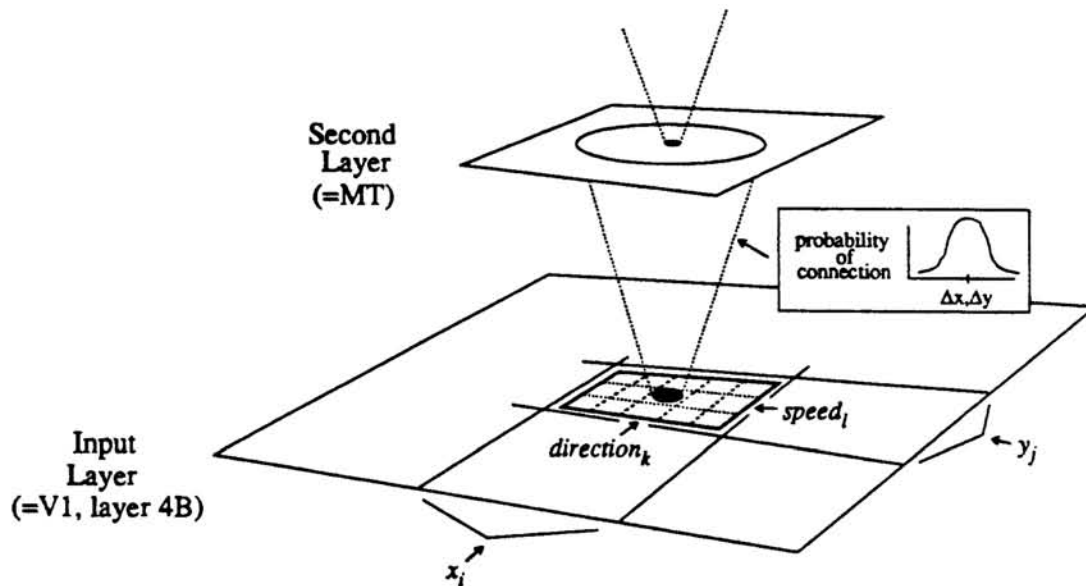

**Figure 2.** Network Architecture

of neurons in MT show a single peak in their direction tuning curves for a plaid that is lined up with the peak for a single grating--i.e., they fire in proportion to the difference between the *true pattern direction* and their best direction (for a bar). These neurons therefore solve the aperture problem presented to them by the local translational motion detectors in layer 4B of V1. The excitatory receptive fields of all MT neurons are much larger than those in V1 as a result of divergence in the V1-MT projection as well as the smaller areal extent of MT compared to V1.

M.E. Sereno (1987) showed using a supervised learning rule that a linear, two layer network can satisfactorily solve the aperture problem characterized above. The present task was to see if unsupervised learning might suffice. A simple caraicature of the V1-to-MT projection was constructed. At each x-y location in the first layer of the network, there are a set of units tuned to a range of local directions and speeds. The input layer thus has four dimensions. The sample network illustrated above (Fig. 2) has 5 different directions and 3 speeds at each x-y location. Input units are granted tuning curves resembling those found for neurons in layer 4B of

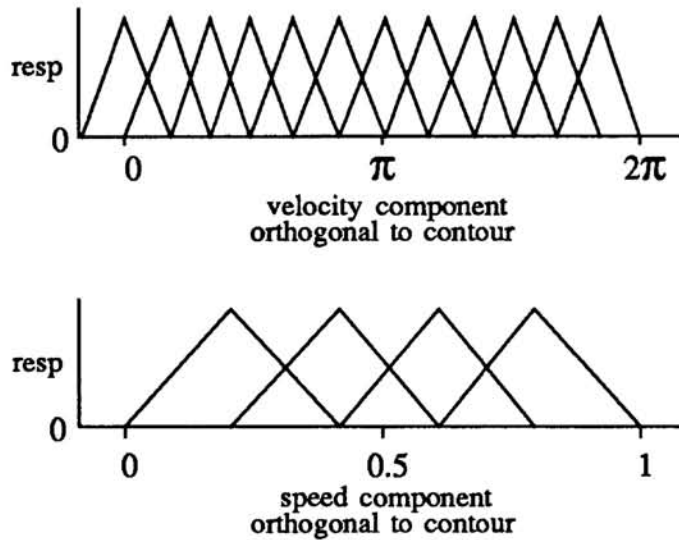

**Figure 3.** Excitatory Tuning
Curves (1st layer)

area V1. The tuning curves are linear, with half-height overlap for both direction
and speed (see Fig. 3--for 12 directions and 4 speeds), and direction and speed tuning
interact linearly. Inhibition is either tuned or untuned (see Fig. 4), and scaled to
balance excitation. Since direction tuning wraps around, there is a trough in the
tuned inhibition condition. Speed tuning does not wrap around. The relative effect
of direction and speed tuning in the output of first layer units is set by a parameter.

As with Linsker, the probability that a unit in the first layer will connect with a
unit in the second layer falls off as a gaussian centered on the retinotopically

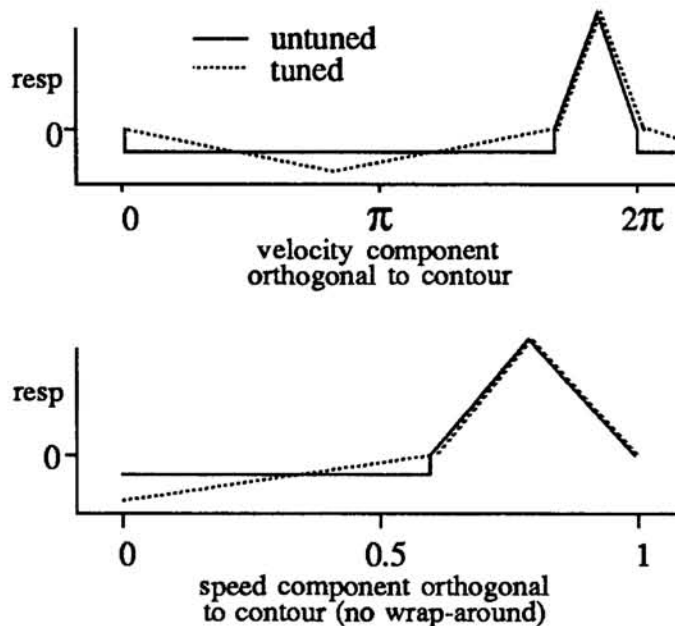

**Figure 4.** Tuned *vs.* Untuned Inhibition

equivalent point in the second layer (see Fig. 2). New random numbers are drawn to generate the divergent gaussian projection pattern for each first layer unit (i.e., all of the units at a single x-y location have different, overlapping projection patterns). There are no local connections within a layer.

The network update rule is similar to that of Linsker except that there is no term like a decay of synaptic weights $(k_1)$ and no offset parameter for the correlations $(k_2)$. Also, all of the units in each layer are modeled explicitly. The activation, $v_j$, for each unit is a linear weighted sum of its $u_i$ inputs, scaled by $\alpha$, and clipped to a maximum or minimum value:

$$v_j = \begin{cases} \alpha \sum u_i w_{ij} \\ \\ v_{max,min} \end{cases}$$

Weights are also clipped to maximum and minimum values. The change in each weight, $\Delta w_{ij}$, is a simple fraction, $\delta$, of the product of the pre- and post-synaptic values:

$$\Delta w_{ij} = \delta u_i v_j$$

## RESULTS

The network is trained with a set of fullfield texture movements. Each stimulus consists of a set of randomly oriented contours--one at each x-y point--all moving in the same, randomly chosen pattern direction. A typical stimulus is drawn in figure 5 as the set of component motions visible to neurons in V1 (i.e., direction components perpendicular to the local contour); the local speed component varies as the cosine of the angle between the pattern direction and the perpendicular to the local contour. The single component motion at each point is run through the first layer tuning curves. The response of the input layer to such a pattern is shown in Figure 6. Each rectangular box represents a single x-y location, containing 48 units

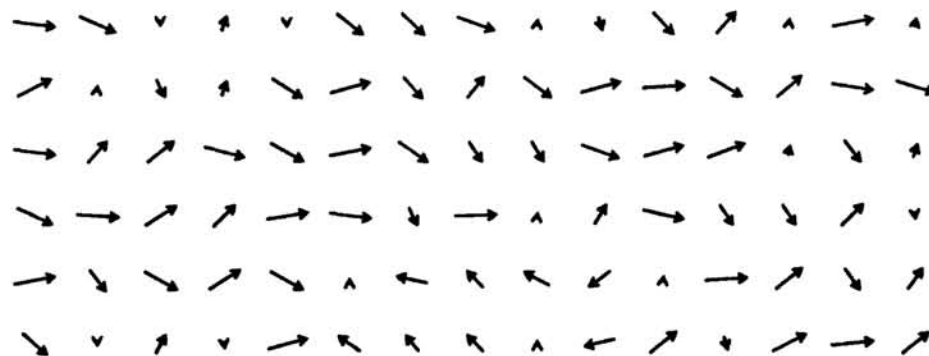

**Figure 5.** Local Component Motions from a Training
Stimulus (pattern direction is toward right)

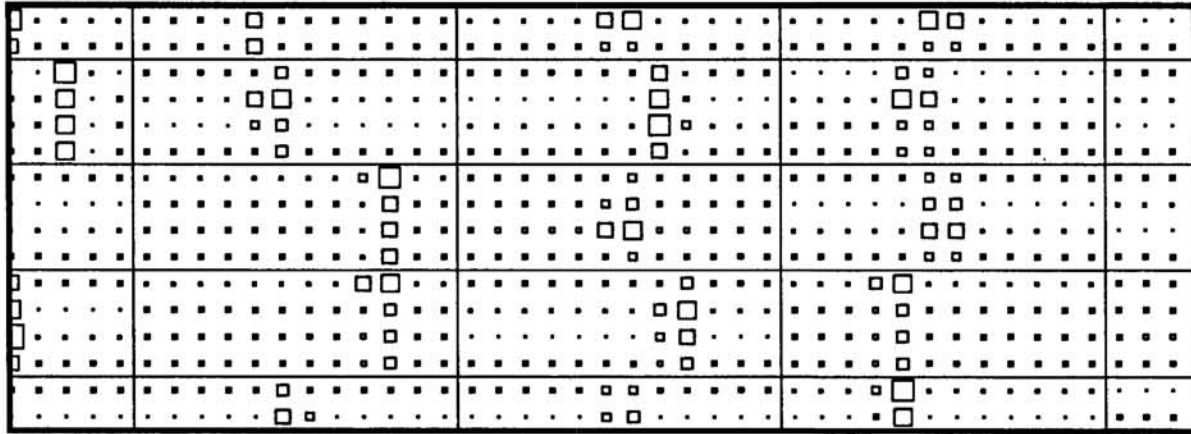

**Figure 6.** Output of Portion of First Layer to a
Training Stimulus (untuned inhibition)

tuned to different combinations of direction and speed (12 directions run horizontally and 4 speeds run vertically). Open and filled squares indicate positive and negative outputs. Inhibition is untuned here. The hebb sensitivity, $\delta$, was set so that 1,000 such patterns could be presented before most weights saturated at maximum values. Weights intially had small random values drawn from a flat distribution centered around zero. The scale parameter for the weighted sum, $\alpha$, was set low enough to prevent second layer units from saturating all the time. In Figure 6, direction tuning is 2.5 times as important as speed tuning in determining the output of a unit.

Selectivity of second layer units for pattern direction was examined both before and after training using four stimulus conditions: 1) *grating*--contours perpendicular to pattern direction, 2) *random grating*--contours randomly oriented with respect to pattern direction (same as the training condition), 3) *plaid*--contours oriented 45 or 67 degrees from perpendicular to pattern direction, 4) *random plaid*--contours randomly oriented, but avoiding angles nearly perpendicular to pattern direction. The pre-training direction tuning curves for the grating conditions usually showed some weak direction selectivity. Pre-training direction tuning curves for the plaid conditions, however, were often twin-peaked, exhibiting pattern component responses displaced to either side of the grating peak. After training, by contrast, the direction tuning peaks in all test conditions were single and sharp, and the plaid condition peaks were usually aligned with the grating peaks.

An example of the weights onto a mature pattern direction selective unit is shown in Figure 7. As before, each rectangular box contains 48 units representing one point in x-y space of the input layer (the tails of the 2-D gaussian are cropped in this illustration), except that the black and white boxes now represent negative and positive weights onto a single second layer unit. Within each box, 12 directions run horizontally and 4 speeds run vertically. The peaks in the direction tuning curves for gratings and 135 degree plaids for this unit were sharp and aligned.

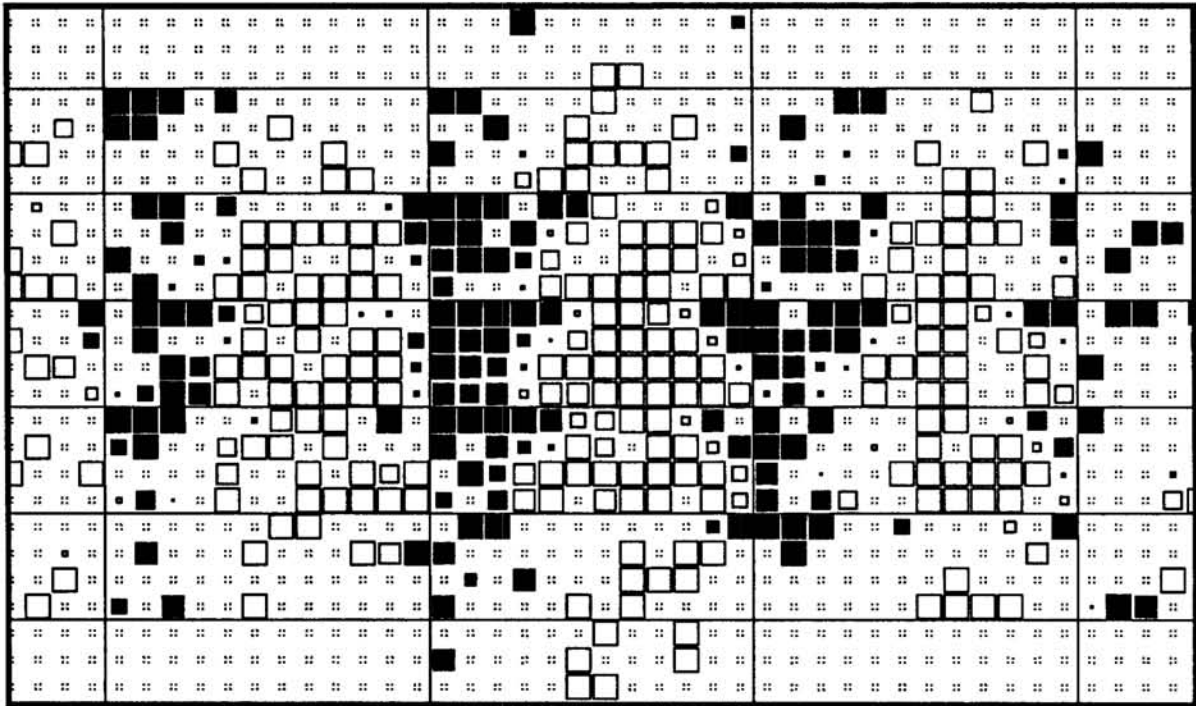

**Figure 7.** Mature Weights Onto Pattern
Direction-Selective Unit

Pattern direction selective units such as this comprised a significant fraction of the second layer when direction tuning was set to be 2 to 4 times as important as speed tuning in determining the output of first layer units. Post-training weight structures under these conditions actually formed a continuum--from units with component direction selectivity, to units with pattern direction selectivity, to units with component speed selectivity. Not surprisingly, varying the relative effects of direction and speed in the V1 tuning curves generated more direction-tuned-only or speed-tuned-only units. In all conditions, units showed clear boundaries between maximum and minimum weights in the direction-speed subspace each x-y point, and a single best direction. The location of these boundaries was always correlated across different x-y input points. Most units showing unambiguous pattern direction selectivity were characterized by two oppositely sloping diagonal boundaries between maximum and minimum weights in direction-speed subspace (see e.g., Fig. 7).

The stimuli used to train the network above--fullfield movments of a rigid texture field of randomly oriented contours--are unnatural; generally, there may be one or more objects in the field moving in different directions and at different speeds than the surround. Weight distributions needed to solve the aperture problem appear when the network is trained on occluding moving objects against moving backgrounds (object and background velocities chosen randomly on each trial), as long as the object is made small or large relative to the receptive field size. The solution breaks down when the moving objects occupy a significant fraction of the area of a second layer receptive field.

For comparison, the network was also trained using two different kinds of noise stimuli. In the first condition (*unit* noise), each new stimulus consisted of random input values on each input unit. With other network parameters held the same, the typical mature weight pattern onto a second layer unit showed an intimate intermixture of maximum and minimum weights in the direction-speed subspace at each x-y location. In the second condition (*direction* noise), each new stimulus consisted of a random direction at each x-y location. The mature weight patterns now showed continuous regions of all-maximum or all-minimum weights in the speed-direction supspace at each x-y point. In contrast to the situation with fullfield texture movement stimuli, however, the best directions at each of the x-y points providing input to a given unit were uncorrelated. In addition, multiple best directions at a single x-y point sometimes appeared.

## DISCUSSION

This simple model suggests that it may be possible to learn the solution to the aperture problem for pattern motion using only biologically realistic unsupervised learning and minimally structured motion fields. Using a similar network architecture, M.E. Sereno had previously shown that supervised learning on the problem of detecting pattern motion direction from local cues leads to the emergence of chevron shaped weight structures in direction-speed space (M.E. Sereno, 1986). The weight structures generated here are similar except that the inside or outside of the chevron is filled in, and upside-down chevrons are more common. This results in decreased selectivity to pattern speed in the second layer.

The model needs to be extended to more complex motion correlations in the input-- e.g., rotation, dilation, shear, multiple objects, flexible objects. MT in primates does not respond selectively to rotation or dilation, while its target area MST does. Thus, biological estimates of rotation and dilation are made in two stages--rotation and dilation are not detected locally, but instead constructed from estimates of local translation. Higher layers in the present model may be able to learn interesting 'second-order' things about rotation, dilation, segmentation, and transparency.

The real primate visual system, of course, has a great many more parts than this model. There are a large number of interconnected cortical visual areas--perhaps as many as 25. A substantial portion of the 600 possible between-area connections may be present (for review, see M.I. Sereno, 1988). There are at least 6 map-like visual structures, and several more non-retinotopic visual structures in the thalamus (beyond the dLGN) that interconnect with the cortical visual areas. Each visual cortical area then has its own set of layers and interlayer connections. The most unbiological aspect of this model is the lack of time and the crude methods of gain control (clipped synaptic weights and input/output functions). Future models should employ within-area connections and time-dependent hebb rules.

Making a biologically realistic model of intermediate and higher level visual processing is difficult since it ostensibly requires making a biologically realistic model of earlier, yet often not less complex stations in the system--e.g., the retina,

dLGN, and layer 4C of primary visual cortex in the present case. One way to avoid having to model all of the stations up to the one of interest is to use physiological data about how the earlier stations respond to various stimuli, as was done in the present model. This shortcut is applicable to many other problems in modeling the visual system. In order for this to be most effective, physiologists and modelers need to cooperate in generating useful libraries of response profiles to arbitrary stimuli. Many stimulus parameters interact, often nonlinearly, to produce the final output of a cell. In the case of simple moving stimuli in V1 and MT, we minimally need to know the interaction between stimulus size, stimulus speed, stimulus direction, surround speed, surround direction, and x-y starting point of the movement relative to the classical excitatory receptive field. Collecting this many response combinations from single cells requires faster serial presentation of stimuli is customary in visual physiology experiments. There is no obvious reason, however, why the rate of stimulus presentation need be any less than the rate at which the visual system normally operates--namely, 3-5 new views per second.

Also, we need to get a better understanding of the 'stimulus set'. The very large set of stimuli on which the real visual system is trained (millions of views) is still very poorly characterized. It would be worthwhile and practical, nevertheless, to collect a naturalistic corpus of perhaps 1000 views (several hours of viewing).

## Acknowledgements

I thank M.E. Sereno and U. Wehmeier for discussions and comments. Supported by NIH grant F32 EY05887. Networks and displays were constructed on the Rochester Connectionist Simulator.

## References

B.K.P. Horn & B.G. Schunck. Determining optical flow. *Artif. Intell.*, **17**, 185-203 (1981).

J. Hyvarinen. *The Parietal Cortex*. Springer-Verlag (1984).

R. Linsker. From basic network principles to neural architecture: emergence of orientation-selective cells. *Proc. Nat. Acad. Sci.* **83**, 8390-8394 (1986).

J.A. Movshon, E.H. Adelson, M.S. Gizzi & W.T. Newsome. Analysis of moving visual patterns. In *Pattern Recognition Mechanisms*. Springer-Verlag, pp. 117-151 (1985).

M.E. Sereno. Modeling stages of motion processing in neural networks. *Proc. 9th Ann. Conf. Cog. Sci. Soc.* pp. 405-416 (1987).

M.I. Sereno. The visual system. In I.W.v. Seelen, U.M. Leinhos, & G. Shaw (eds.), *Organization of Neural Networks*. VCH, pp. 176-184 (1988).

R.V. Spitz, J. Stiles-Daves & R.M. Siegel. Infant perception of rotation from rigid structure-from-motion displays. *Neurosci. Abstr.* **14**, 1244 (1988).